# Statistical Models of Conditioning

**Peter Dayan\***
Brain & Cognitive Sciences
E25-210 MIT
Cambridge, MA 02139

**Theresa Long**
123 Hunting Cove
Williamsburg, VA 23185

## Abstract

Conditioning experiments probe the ways that animals make predictions about rewards and punishments and use those predictions to control their behavior. One standard model of conditioning paradigms which involve many conditioned stimuli suggests that individual predictions should be added together. Various key results show that this model fails in some circumstances, and motivate an alternative model, in which there is attentional selection between different available stimuli. The new model is a form of mixture of experts, has a close relationship with some other existing psychological suggestions, and is statistically well-founded.

## 1 Introduction

Classical and instrumental conditioning experiments study the way that animals learn about the causal texture of the world (Dickinson, 1980) and use this information to their advantage. Although it reached a high level of behavioral sophistication, conditioning has long since gone out of fashion as a paradigm for studying learning in animals, partly because of the philosophical stance of many practitioners, that the neurobiological implementation of learning is essentially irrelevant. However, more recently it has become possible to study how conditioning phenomena are affected by particular lesions or pharmacological treatments to the brain (*eg* Gallagher & Holland, 1994), and how particular systems, during simple learning tasks, report information that is consistent with models of conditioning (Gluck & Thompson, 1987; Gabriel & Moore, 1989).

In particular, we have studied the involvement of the dopamine (DA) system in the ventral tegmental area of vertebrates in reward based learning (Montague *et al*, 1996; Schultz *et al*, 1997). The activity of these cells is consistent with a model in which they report a temporal difference (TD) based prediction error for reward

(Sutton & Barto, 1981; 1989). This prediction error signal can be used to learn correct predictions and also to learn appropriate actions (Barto, Sutton & Anderson, 1983). The DA system is important since it is crucially involved in normal reward learning, and also in the effects of drugs of addiction, self stimulation, and various neural diseases.

The TD model is consistent with a whole body of experiments, and has even correctly anticipated new experimental findings. However, like the Rescorla-Wagner (RW; 1972) or delta rule, it embodies a particular additive model for the net prediction made when there are multiple stimuli. Various sophisticated conditioning experiments have challenged this model and found it wanting. The results support competitive rather than additive models. Although *ad hoc* suggestions have been made to repair the model, none has a sound basis in appropriate prediction. There is a well established statistical theory for competitive models, and it is this that we adopt.

In this paper we review existing evidence and theories, show what constraints a new theory must satisfy, and suggest and demonstrate a credible candidate. Although it is based on behavioral data, it also has direct implications for our neural theory.

## 2 Data and Existing Models

Table 1 describes some of the key paradigms in conditioning (Dickinson, 1980; Mackintosh, 1983). Although the collection of experiments may seem rather arcane (the standard notation is even more so), in fact it shows exactly the basis behind the key capacity of animals in the world to predict events of consequence. We will extract further biological constraints implied by these and other experiments in the discussion.

In the table, $l$ (light) and $s$ (tone) are potential predictors (called *conditioned stimuli* or CSs), of a consequence, $r$, such as the delivery of a reward (called an *unconditioned stimulus* or US). Even though we use TD rules in practice, we discuss some of the abstract learning rules without much reference to the detailed time course of trials. The same considerations apply to TD.

In Pavlovian conditioning, the light acquires a positive association with the reward in a way that can be reasonably well modeled by:

$$\Delta w_l(t) = \alpha_l(t)(r(t) - w_l(t))l(t), \tag{1}$$

where $l(t) \in \{0, 1\}$ represents the presence of the light in trial $t$ ($s(t)$ will similarly represent the presence of a tone), $w_l(t)$ (we will often drop the index $t$) represents the strength of the expectation about the delivery of reward $r(t)$ in trial $t$ if the light is also delivered, and $\alpha_l(t)$ is the learning rate. This is just the delta rule. It also captures well the probabilistic contingent nature of conditioning – for binary $r(t) \in \{0, 1\}$, animals seem to assess $\epsilon_l = \mathcal{P}[r(t)|l(t)=1] - \mathcal{P}[r(t)|l(t)=0]$, and then only expect reward following the light (in the model, have $w_l > 0$) if $\epsilon_l > 0$.

Pavlovian conditioning is easy to explain under a whole wealth of rules. The trouble comes in extending equation 1 to the case of multiple predictors (in this paper we consider just two). The other paradigms in table 1 probe different aspects of this. The one that is most puzzling is (perversely) called *downwards unblocking* (Holland, 1988). In a first set of trials, an association is established between the light and two presentations of reward separated by a few ($u$) seconds. In a second set, a tone is included with the light, but the second reward is dropped. The animal amasses *less* reward in conjunction with the tone. However, when presented with the tone

|   | Name | Set 1 | Set 2 | Test |
|---|------|-------|-------|------|
| 1 | Pavlovian | | $l \to r$ | $l \looparrowright r$ |
| 2 | Overshadowing | | $l + s \to r$ | $\left\{ \begin{array}{c} l \looparrowright r^{\frac{1}{2}} \\ s \looparrowright r^{\frac{1}{2}} \end{array} \right\}$ |
| 3 | Inhibitory | | $\left\{ \begin{array}{c} l \to r \\ l + s \to \cdot \end{array} \right\}$ | $s \looparrowright \bar{r}$ |
| 4 | Blocking | $l \to r$ | $l + s \to r$ | $s \looparrowright \cdot$ |
| 5 | Upwards unblocking | $l \to r$ | $l + s \to r\Delta_u r$ | $s \looparrowright r$ |
| .6 | Downwards unblocking | $l \to r\Delta_u r$ | $l + s \to r$ | $s \looparrowright \pm r$ |

Table 1: Paradigms. Sets 1 and 2 are separate sets of learning trials, which are continued until convergence. Symbols $l$ and $s$ indicate presentation of lights and tones as potential predictors. The $\looparrowright$ in the test set indicates that the associations of the predictors are tested, producing the listed results. In overshadowing, association with the reward can be divided between the light and the sound, indicated by $r^{\frac{1}{2}}$. In some cases overshadowing favours one stimulus at the complete expense of the other; and at the end of very prolonged training, all effects of overshadowing can disappear. In blocking, the tone makes no prediction of $r$. In set 2 of inhibitory conditioning, the two types of trials are interleaved and the outcome is that the tone predicts the absence of reward. In upwards and downwards unblocking, the $\Delta_u$ indicates that the delivery of two rewards is separated by time $u$. For downwards unblocking, if $u$ is small, then $s$ is associated with the *absence* of $r$; if $u$ is large, then $s$ is associated with the *presence* of $r$.

alone, the animal expects the *presence* rather than the absence of reward. On the face of it, this seems an insurmountable challenge to prediction-based theories. First we describe the existing theories, then we formalise some potential replacements.

One theory (called a US-processing theory) is due to Rescorla & Wagner (RW; 1972), and, as pointed out by Sutton & Barto (1981), is just the delta rule. For RW, the animal constructs a net prediction:

$$V(t) = w_l(t)l(t) + w_s(t)s(t) \tag{2}$$

for $r(t)$, and then changes $\Delta w_l(t) = \alpha_l(t)(r(t) - V(t))l(t)$ (and similarly for $w_s(t)$) using the prediction error $r(t) - V(t)$. Its foundation in the delta rule makes it computationally appropriate (Marr, 1982) as a method of making predictions. TD uses the same additive model in equation 2, but uses $r(t) + V(t+1) - V(t)$ as the prediction error.

RW explains overshadowing, inhibitory conditioning, blocking, and upwards unblocking, but *not* downwards unblocking. In overshadowing, the terminal association between $l$ and $r$ is weaker if $l$ and $s$ are simultaneously trained – this is expected under RW since learning stops when $V(t) = r(t)$, and $w_l$ and $w_s$ will share the prediction. In inhibitory conditioning, the sound comes to predict the absence of $r$. The explanation of inhibitory conditioning is actually quite complicated (Konorski, 1967; Mackintosh, 1983); however RW provides the simple account that $w_l = r$ for the $l \to r$ trials, forcing $w_s = -r$ for the $l+s \to \cdot$ trials. In blocking, the prior association between $l$ and $r$ means that $w_l = r$ in the second set of trials, leading to no learning for the tone (since $V(t) - r(t) = 0$). In upwards unblocking, $w_l = r$ at the start of set 2. Therefore, $r(t) - w_l = r > 0$, allowing $w_s$ to share in the prediction.

As described above, downwards unblocking is the key thorn in the side of RW. Since the TD rule combines the predictions from different stimuli in a similar way,

it also fails to account properly for downwards unblocking. This is one reason why it is *incorrect* as a model of reward learning.

The class of theories (called CS-processing theories) that is alternative to RW does not construct a net prediction $V(t)$, but instead uses equation 1 for all the stimuli, only changing the learning rates $\alpha_l(t)$ and $\alpha_s(t)$ as a function of the conditioning history of the stimuli (*eg* Mackintosh, 1975; Pearce & Hall, 1980; Grossberg, 1982). A standard notion is that there is a competition between different stimuli for a limited capacity learning processor (Broadbent, 1958; Mackintosh, 1975; Pearce & Hall, 1980), translating into competition between the learning rates. In blocking, nothing unexpected happens in the second set of trials and equally, the tone does not predict anything novel. In either case $\alpha_s$ is set to $\sim 0$ and so no learning happens. In these models, downwards unblocking now makes qualitative sense: the surprising consequences in set 2 can be enough to set $\alpha_s \gg 0$, but then learning according to equation 1 can make $w_s > 0$. Whereas Mackintosh's (1975) and Pearce and Hall's (1980) models only consider competition between the stimuli for *learning*, Grossberg's (1982) model incorporates competition during *representation*, so the net prediction on a trial is affected by competitive interactions between the stimuli. In essence, our model provides a statistical formalisation of this insight.

## 3  New Models

From the previous section, it would seem that we have to abandon the computational basis of the RW and TD models in terms of making collective predictions about the reward. The CS-processing models do not construct a net prediction of the reward, or say anything about how possibly conflicting information based on different stimuli should be integrated. This is a key flaw – doing anything other than well-founded prediction is likely to be maladaptive. Even quite successful pre-synaptic models, such as Grossberg (1982), do not justify their predictions.

We now show that we can take a different, but still statistically-minded approach to combination in which we specify a parameterised probability distribution $\mathcal{P}[r(t)|s(t), l(t)]$ and perform a form of maximum likelihood (ML) inference, updating the parameters to maximise this probability over the samples. Consider three natural models of $\mathcal{P}[r(t)|s(t), l(t)]$:

$$\mathcal{P}_G[r(t)|s(t), l(t)] = \mathcal{N}[w_l l(t) + w_s s(t), \sigma^2] \tag{3}$$

$$\mathcal{P}_M[r(t)|s(t), l(t)] = \pi_l(t)\mathcal{N}[w_l, \sigma^2] + \pi_s(t)\mathcal{N}[w_s, \sigma^2] + \bar{\pi}(t)\mathcal{N}[\bar{w}, \tau^2] \tag{4}$$

$$\mathcal{P}_J[r(t)|s(t), l(t)] = \mathcal{N}[w_l \pi_l(t) l(t) + w_s \pi_s(t) s(t), \sigma^2] \tag{5}$$

where $\mathcal{N}[\mu, \sigma^2]$ is a normal distribution, with mean $\mu$ and variance $\sigma^2$. In the latter two cases, $0 \leq \pi_l(t) + \pi_s(t) \leq 1$, implementing a form of competition between the stimuli, and $\pi_*(t) = 0$ if stimulus $*$ is not presented. In equation 4, $\mathcal{N}[\bar{w}, \tau^2]$ captures the background expectation if neither the light nor the tone wins, and $\bar{\pi}(t) = 1 - \pi_l(t) - \pi_s(t)$. We will show that the data argue against the first two and support the third of these models.

**Rescorla-Wagner:** $\mathcal{P}_G[r(t)|s(t), l(t)]$

The RW rule is derived as ML inference based on equation 3. The only difference is the presence of the variance, $\sigma^2$. This is useful for capturing the partial reinforcement effect (see Mackintosh, 1983), in which if $r(t)$ is corrupted by substantial noise (*ie* $\sigma^2 \gg 0$), then learning to $r$ is demonstrably slower. As we discussed above,

downwards unblocking suggests that animals are not using $\mathcal{P}_G[r(t)|s(t), l(t)]$ as the basis for their predictions.

**Competitive mixture of experts:** $\mathcal{P}_M[r(t)|s(t), l(t)]$

$\mathcal{P}_M[r(t)|s(t), l(t)]$ is recognisable as the generative distribution in a mixture of Gaussians model (Nowlan, 1991; Jacobs *et al*, 1991b). Key in this model are the mixing proportions $\pi_l(t)$ and $\pi_s(t)$. Online variants of the E phase of the EM algorithm (Dempster *et al*, 1977) compute posterior responsibilities as $q_l(t) + q_s(t) + \bar{q}(t) = 1$, where $q_l(t) \propto \pi_l(t)e^{-(r(t)-w_l l(t))^2/2\sigma^2}$ (and similarly for the others), and then perform a partial M phase as

$$\Delta w_l(t) \propto (r(t) - w_l(t))q_l(t) \qquad \Delta w_s(t) \propto (r(t) - w_s(t))q_s(t) \qquad (6)$$

which has just the same character as the presynaptic rules (depending on how $\pi_l(t)$ is calculated). As in the mixture of experts model, each expert (each *stimulus* here) that seeks to predict $r(t)$ (*ie* each stimulus $*$ for which $q_*(t) \neq 0$) has to predict the whole of $r(t)$ by itself. This means that the model can capture downwards unblocking in the following way. The absence of the second $r$ in the second set of trials forces $\pi_s(t) > 0$, and, through equation 6, this in turn means that the tone will come to predict the presence of the first $r$. The time $u$ between the rewards can be important because of temporal discounting. This means that there are sufficiently large values of $u$ for which the inhibitory effect of the absence of the second reward will be dominated. Note also that the *expected* reward based on $l(t)$ and $s(t)$ is the sum

$$\pi_l(t)w_l l(t) + \pi_s(t)w_s s(t) + \bar{\pi}(t)\bar{w} \qquad (7)$$

Although the net prediction given in equation 7 is indeed based on all the stimuli, it does not directly affect the course of learning. This means that the model has difficulty with inhibitory conditioning. The trouble with inhibitory conditioning is that the model cannot use $w_s < 0$ to counterbalance $w_l > 0$ – it can at best set $w_s = 0$, which is experimentally inaccurate. Note, however, this form of competition bears some interesting similarities with comparator models of conditioning (see Miller & Matzel, 1989). It also has some problems in explaining overshadowing, for similar reasons.

**Cooperative mixture of experts:** $\mathcal{P}_J[r(t)|s(t), l(t)]$

The final model $\mathcal{P}_J[r(t)|s(t), l(t)]$ is just like the mixture model that Jacobs *et al* (1991a) suggested (see also Bordley, 1982). One statistical formulation of this model considers that, independently,

$$P[w_l(t)|r] = \mathcal{N}[r, \rho_l^{-1}(t)] \qquad P[w_s(t)|r] = \mathcal{N}[r, \rho_s^{-1}(t)]$$

where $\rho_l(t)$ and $\rho_s(t)$ are inverse variances. This makes

$$\sigma^2 = (\rho_l(t) + \rho_s(t))^{-1} \qquad \pi_l(t) = \rho_l(t)\sigma^2 \qquad \pi_s(t) = \rho_s(t)\sigma^2.$$

Normative learning rules should emerge from a statistical model of uncertainty in the world. Short of such a model, we used:

$$\Delta w_l = \alpha_w \frac{\pi_l(t)}{\rho_l(t)}\delta(t) \qquad \Delta \rho_l = \alpha_\rho \rho_l \left( \frac{1}{\delta(t)^2 + 0.1} - \frac{1}{\sigma^2} \right)$$

where $\delta(t) = r(t) - \pi_l(t)w_l(t) - \pi_s(t)w_s(t)$ is the prediction error; the $1/\rho_l(t)$ term in changing $w_l$ makes learning *slower* if $w_l$ is more certainly related to $r$ (*ie* if $\rho_l(t)$ is greater); the 0.1 substitutes for background noise; if $\delta^2(t)$ is too large, then $\rho_l + \rho_s$

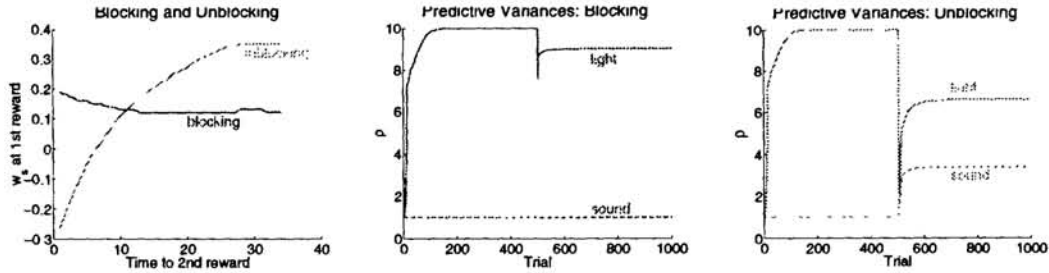

Figure 1: Blocking and downwards unblocking with 5 steps to the first reward; and a variable number to the second. Here, the discount factor $\gamma = 0.9$, and $\alpha_w = 0.5$, $\alpha_\rho = 0.02$, $\mu = 0.75$. For blocking, the second reward remains; for unblocking it is removed after 500 trials. a) The terminal weight for the sound after learning – for blocking it is always small and positive; for downwards unblocking, it changes from negative at small $\Delta_u$ to positive at large $\Delta_u$. b,c) Predictive variances $\rho_l(t)$ and $\rho_s(t)$. In blocking, although there is a small change when the sound is introduced because of additivity of the variances, learning to the sound is substantially prevented. In downwards unblocking, the surprise omission of the second reward makes the sound associable and unblocks learning to it.

is shared out in proportion of $\rho_l^\mu$ to capture the insight that there can be dramatic changes to variabilities; and the variabilities are bottom-limited.

Figure 1 shows the end point and course of learning in blocking and downwards unblocking. Figure 1a confirms that the model captures downwards unblocking, making the terminal value of $w_s$ negative for short separations between the rewards and positive for long separations. By comparison, in the blocking condition, for which both rewards are always presented, $w_s$ is always small and positive. Figures 1b,c show the basis behind this behaviour in terms of $\rho_l(t)$ and $\rho_s(t)$. In particular, the heightened associability of the sound in unblocking following the prediction error when the second reward is removed accounts for the behavior.

As for the mixture of experts model (and also for comparator models), the presence of $\pi_l(t)$ and $\pi_s(t)$ makes the explanation of inhibitory conditioning and overshadowing a little complicated. For instance, if the sound is associable ($\rho_s(t) \gg 0$), then it can seem to act as a conditioned inhibitor even if $w_s = 0$. Nevertheless, unlike the mixture of experts model, the fact that learning is based on the joint prediction makes true inhibitory conditioning possible.

## 4  Discussion

Downwards unblocking may seem like an extremely abstruse paradigm with which to refute an otherwise successful and computationally sound model. However, it is just the tip of a conditioning iceberg that would otherwise sink TD. Even in other reinforcement learning applications of TD, there is no *a priori* reason why predictions should be made according to equation 2 – the other statistical models in equations 4 and 5 could also be used. Indeed, it is easy to generate circumstances in which these more competitive models will perform better. For the neurobiology, experiments on the behavior of the DA system in these conditioning tasks will help specify the models further.

The model is incomplete in various important ways. First, it makes no distinction between preparatory and consumatory conditioning (Konorski, 1967). There is evidence that the predictions a CS makes about the *affective* value of USs fall in a different class from the predictions it makes about the actual USs that appear.

For instance, an inhibitory stimulus reporting the absence of expected delivery of food can block learning to the delivery of shock, implying that aversive events form a single class. The affective value forms the preparatory aspect, is likely what is reported by the DA cells, and perhaps controls orienting behavior, the characteristic reaction of animals to the conditioned stimuli that may provide an experimental handle on the attention they are paid. Second, the model does not use opponency (Konorski, 1967; Solomon & Corbit, 1974; Grossberg, 1982) to handle inhibitory conditioning. This is particularly important, since the dynamics of the interaction between the opponent systems may well be responsible for the importance of the delay $u$ in downwards unblocking. Serotonin is an obvious candidate as an opponent system to DA (Montague *et al* 1996). We also have not specified a substrate for the associabilities or the attentional competition – the DA system itself may well be involved. Finally, we have not specified an overall model of how the animal might expect the contingency of the world to change over time – which is key to the statistical justification of appropriate learning rules.

## Footnotes

\*This work was funded by the Surdna Foundation.

# References

[1] Barto, AG, Sutton, RS & Anderson, CW (1983). *IEEE Transactions on Systems, Man, and Cybernetics,* **13**, pp 834-846.

[2] Bordley, RF (1982). *Journal of the Operational Research Society,* **33**, 171-174.

[3] Broadbent, DE (1958). *Perception and Communication.* London: Pergamon.

[4] Buhusi, CV & Schmajuk, NA. *Hippocampus,* **6**, 621-642.

[5] Dempster, AP, Laird, NM & Rubin, DB (1977). *Proceedings of the Royal Statistical Society,* **B-39**, 1–38.

[6] Dickinson, A (1980). *Contemporary Animal Learning Theory.* Cambridge, England: Cambridge University Press.

[7] Gabriel, M & Moore, J, editors (1989). *Learning and Computational Neuroscience.* Cambridge, MA: MIT Press.

[8] Gallagher, M & Holland, PC (1994). *PNAS,* **91**, 11771-6.

[9] Gluck, MA & Thompson, RF (1987). *Psychological Reviews,* **94**, 176-191.

[10] Grossberg, S (1982). *Psychological Review,* **89**, 529-572.

[11] Holland, PC (1988). *Journal of Experimental Psychology: Animal Behavior Processes,* **14**, 261-279.

[12] Jacobs, RA, Jordan, MI & Barto, AG (1991). *Cognitive Science,* **15**, 219-250.

[13] Jacobs, RA, Jordan, MI, Nowlan, SJ & Hinton, GE (1991). *Neural Computation,* **3**, 79-87.

[14] Konorski, J (1967). *Integrative Activity of the Brain.* Chicago, Il: Chicago University Press.

[15] Mackintosh, NJ (1975). *Psychological Review,* **82**, 276-298.

[16] Mackintosh, NJ (1983). *Conditioning and Associative Learning.* Oxford, UK: Oxford University Press.

[17] Marr, D (1982). *Vision.* New York, NY: Freeman.

[18] Miller, RR & Matzel, LD (1989). In SB Klein & RR Mowrer, editors, *Contemporary Learning Theories: Pavlovian Conditioning and the Status of Traditional Theory.* Hillsdale, NJ: Lawrence Erlbaum.

[19] Montague, PR, Dayan, P & Sejnowski, TK (1996). *Journal of Neuroscience,* **16**, 1936-1947.

[20] Nowlan, SJ (1991). *Soft Competitive Adaptation: Neural Network Learning Algorithms Based on Fitting Statistical Mixtures.* PhD Thesis, Department of Computer Science, Carnegie-Mellon University.

[21] Pearce, JM & Hall, G (1980). *Psychological Review,* **87**, 532-552.

[22] Rescorla, RA & Wagner, AR (1972). In AH Black & WF Prokasy, editors, *Classical Conditioning II: Current Research and Theory,* pp 64-69. New York, NY: Appleton-Century-Crofts.

[23] Schultz, W, Dayan, P & Montague, PR (1997). *Science,* **275**, 1593-1599.

[24] Solomon, RL & Corbit, JD (1974). *Psychological Review,* **81**, 119-145.

[25] Sutton, RS & Barto, AG (1981). *Psychological Review,* **88** 2, pp 135-170.

[26] Sutton, RS & Barto, AG (1989). In Gabriel & Moore (1989).
